# Multi-Agent Filtering with Infinitely Nested Beliefs

**Luke S. Zettlemoyer**
MIT CSAIL
Cambridge, MA 02139
lsz@csai.mit.edu

**Brian Milch**$^{*}$
Google Inc.
Mountain View, CA 94043
brian@google.com

**Leslie Pack Kaelbling**
MIT CSAIL
Cambridge, MA 02139
lpk@csai.mit.edu

## Abstract

In partially observable worlds with many agents, nested beliefs are formed when agents simultaneously reason about the unknown state of the world and the beliefs of the other agents. The multi-agent filtering problem is to efficiently represent and update these beliefs through time as the agents act in the world. In this paper, we formally define an infinite sequence of nested beliefs about the state of the world at the current time $t$, and present a filtering algorithm that maintains a finite representation which can be used to generate these beliefs. In some cases, this representation can be updated exactly in constant time; we also present a simple approximation scheme to compact beliefs if they become too complex. In experiments, we demonstrate efficient filtering in a range of multi-agent domains.

## 1 Introduction

The existence of nested beliefs is one of the defining characteristics of a multi-agent world. As an agent acts, it often needs to reason about what other agents believe. For instance, a teacher must consider what a student knows to decide how to explain important concepts. A poker agent must think about what cards other players might have — and what cards they might think it has — in order to bet effectively. In this paper, we assume a cooperative setting where all the agents have predetermined, commonly-known policies expressed as functions of their beliefs; we focus on the problem of efficient belief update, or filtering.

We consider the nested filtering problem in multi-agent, partially-observable worlds [6, 1, 9]. In this setting, agents receive separate observations and independently execute actions, which jointly change the hidden state of the world. Since each agent does not get to see the others' observations and actions, there is a natural notion of nested beliefs. Given its observations and actions, an agent can reason not only about the state of the external world, but also about the other agents' observations and actions. It can also condition on what others might have seen and done to compute their beliefs at the next level of nesting. This pattern can be repeated to arbitrary depth.

The multi-agent filtering problem is to efficiently represent and update these nested beliefs through time. In general, an agent's beliefs depend on its entire history of actions and observations. One approach to computing these beliefs would be to remember the entire history, and perform inference to compute whatever probabilities are needed at each time step. But the time required for this computation would grow with the history length. Instead, we maintain a *belief state* that is sufficient for predicting future beliefs and can be approximated to achieve constant-time belief updates.

We begin by defining an infinite sequence of nested beliefs about the current state $s_t$, and showing that it is sufficient for predicting future beliefs. We then present a multi-agent filtering algorithm that maintains a compact representation sufficient for generating this sequence. Although in the worst case this representation grows exponentially in the history length, we show that its size remains constant for several interesting problems. We also describe an approximate algorithm that always

---

$^{*}$This work was done while the second author was at MIT CSAIL.

maintains a constant representation size (and constant-time updates), possibly at the cost of accuracy. In experiments, we demonstrate efficient and accurate filtering in a range of multi-agent domains.

## 2 Related Work

In existing research on partially observable stochastic games (POSGs) and Decentralized POMDPs (DEC-POMDPs) [6, 1, 9], policies are represented as direct mappings from observation histories to actions. That approach removes the need for the agents to perform any kind of filtering, but requires the specification of some particular class of policies that return actions for arbitrarily long histories. In contrast, many successful algorithms for single-agent POMDPs represent policies as functions on belief states [7], which abstract over the specifics of particular observation histories. Gmytrasiewicz and Doshi [5] consider filtering in *interactive POMDPs*. Their approach maintains finitely nested beliefs that are derived from a world model as well as hand-specified models of how each agent reasons about the other agents. In this paper, all of the nested reasoning is derived from a single world model, which eliminates the need for any agent-specific models.

To the best of our knowledge, our work is the first to focus on filtering of infinitely nested beliefs. There has been significant work on infinitely nested beliefs in game theory, where Brandenburger and Dekel [2] introduced the notion of an infinite sequence of finitely nested beliefs. However, they do not describe any method for computing these beliefs from a world model or updating them over time. Another long-standing line of related work is in the epistemic logic community. Fagin and Halpern [3] define labeled graphs called *probabilistic Kripke structures*, and show how a graph with finitely many nodes can define an infinite sequence of nested beliefs. Building on this idea, algorithms have been proposed for answering queries on probabilistic Kripke structures [10] and on influence diagrams that define such structures [8]. However, these algorithms have not addressed the fact that as agents interact with the world over time, the set of observation sequences they could have received (and possibly the set of beliefs they could arrive at) grows exponentially.

## 3 Nested Filtering

In this section, we describe the world model and define the multi-agent filtering problem. We then present a detailed example where a simple problem leads to a complex pattern of nested reasoning.

### 3.1 Partially observable worlds with many agents

We will perform filtering given a multi-agent, decision-theoretic model for acting in a partially observable world.[1] Agents receive separate observations and independently execute actions, which jointly change the state of the world. There is a finite set of states $S$, but the current state $s \in S$ cannot be observed directly by any of the agents. Each agent $j$ has a finite set of observations $O^j$ that it can receive and a finite set of actions $A^j$ that it can execute. Throughout this paper, we will use superscripts and vector notation to name agents and subscripts to indicate time. For example, $a_t^j \in A^j$ is the action for agent $j$ at time $t$; $\vec{a}_t = \langle a_t^i, \ldots, a_t^j \rangle$ is a vector with actions for each of the agents; and $a_{0:t}^j = (a_0^j, \ldots, a_t^j)$ is a sequence of actions for agent $j$ at time steps $0 \ldots t$.

The state dynamics is defined by a distribution $p_0(s)$ over initial states and a transition distribution $p(s_t | s_{t-1}, \vec{a}_{t-1})$ that is conditioned on the previous state $s_{t-1}$ and the action vector $\vec{a}_{t-1}$. For each agent $j$, observations are generated from a distribution $p(o_t^j | s_t, \vec{a}_{t-1})$ conditioned on the current state and the previous joint action. Each agent $j$ sees only its own actions and observations. To record this information, it is useful to define a *history* $h_{0:t}^j = (a_{0:t-1}^j, o_{1:t}^j)$ for agent $j$ at time $t$. A policy is a distribution $\pi^j(a_t^j | h_{0:t}^j)$ over the actions agent $j$ will take given this history. Together, these distributions define the joint world model:

$$p(s_{0:t}, \vec{h}_{0:t}) = p_0(s_0) \prod_{i=0}^{t-1} \vec{\pi}(\vec{a}_i | \vec{h}_{0:i}) p(s_{i+1} | s_i, \vec{a}_i) p(\vec{o}_{i+1} | s_{i+1}, \vec{a}_i) \tag{1}$$

where $\vec{\pi}(\vec{a}_t | \vec{h}_{0:t}) = \prod_j \pi^j(a_t^j | h_{0:t}^j)$ and $p(\vec{o}_{t+1} | s_{t+1}, \vec{a}_t) = \prod_j p(o_{t+1}^j | s_{t+1}, \vec{a}_t)$.

## 3.2 The nested filtering problem

In this section, we describe how to compute infinitely nested beliefs about the state at time $t$. We then define a class of policies that are functions of these beliefs. Finally, we show that the current nested belief for an agent $i$ contains all of the information required to compute future beliefs. Throughout the rest of this paper, we use a minus notation to define tuples indexed by all but one agent. For example, $h_{0:t}^{-i}$ and $\pi^{-i}$ are tuples of histories and policies for all agents $k \neq i$.

We define infinitely nested beliefs by presenting an infinite sequence of finitely nested beliefs. For each agent $i$ and nesting level $n$, the belief function $B^{i,n} : h_{0:t}^i \rightarrow b_t^{i,n}$ maps the agent's history to its $n$th-level beliefs at time $t$. The agent's *zeroth-level* belief function $B^{i,0}(h_{0:t}^i)$ returns the posterior distribution $b_t^{i,0} = p(s_t|h_{0:t}^i)$ over states given the input history, which can be computed from Eq. 1:

$$B^{i,0}(h_{0:t}^i) = p(s_t|h_{0:t}^i) \propto \sum_{s_{0:t-1}, h_{0:t}^{-i}} p(s_{0:t}, \vec{h}_{0:t}).$$

Agent $i$'s *first-level* belief function $B^{i,1}(h_{0:t}^i)$ returns a joint distribution on $s_t$ and the zeroth-level beliefs of all the other agents (what the other agents believe about the state of the world). We can compute the tuple of zeroth-level beliefs $b_t^{-i,0}$ for all agents $k \neq i$ by summing the probabilities of all histories $h_{0:t}^{-i}$ that lead to these beliefs (that is, such that $b_t^{-i,0} = B^{-i,0}(h_{0:t}^{-i})$):

$$B^{i,1}(h_{0:t}^i) = p(s_t, b_t^{-i,0}|h_{0:t}^i) \propto \sum_{s_{0:t-1}, h_{0:t}^{-i}} p(s_{0:t}, \vec{h}_{0:t})\delta(b_t^{-i,0}, B^{-i,0}(h_{0:t}^{-i})).$$

The delta function $\delta(\cdot, \cdot)$ returns one when its arguments are equal and zero otherwise.

For level $n$, $B^{i,n}(h_{0:t}^i)$ returns a distribution over states and level $n-1$ beliefs for the other agents. For example, at level 2, the function returns a joint distribution over: the state, what the other agents believe about the state, and what they believe others believe. Again, these beliefs are computed by summing over histories for the other agents that lead to the appropriate level $n-1$ beliefs:

$$B^{i,n}(h_{0:t}^i) = p(s_t, b_t^{-i,n-1}|h_{0:t}^i) \propto \sum_{s_{0:t-1}, h_{0:t}^{-i}} p(s_{0:t}, \vec{h}_{0:t})\delta(b_t^{-i,n-1}, B^{-i,n-1}(h_{0:t}^{-i})).$$

Note that for all nesting levels $n$, $B^{i,n}(h_{0:t}^i)$ is a discrete distribution. There are only finitely many beliefs each agent $k$ could hold at time $t$ — each arising from one of the possible histories $h_{0:t}^k$.

Define $b_t^{i,*} = B^{i,*}(h_{0:t}^i)$ to be the infinite sequence of nested beliefs generated by computing $B^{i,n}(h_{0:t}^i)$ for $n = 0, 1, \ldots$. We can think of $b_t^{i,*}$ as a belief state for agent $i$, although not one that can be used directly by a filtering algorithm. We will assume that the policies $\pi^i$ are represented as functions of these belief states: that is, $\pi^i(a_t^i|b_t^{i,*})$ can be thought of as a procedure that looks at arbitrary parts of the infinite sequence $b_t^{i,*}$ and returns a distribution over actions. We will see examples of this type of policy in the next section. Under this assumption, $b_t^{i,*}$ is a sufficient statistic for predicting future beliefs in the following sense:

**Proposition 1** *In a model with policies* $\pi^j(a_t^j|b_t^{j,*})$ *for each agent $j$, there exists a* belief estimation function $BE$ s.t. $\forall a_{0:t-1}^i, o_{1:t}^i, a_t^i, o_{t+1}^i$ . $B^{i,*}(a_{0:t}^i, o_{1:t+1}^i) = BE(B^{i,*}(a_{0:t-1}^i, o_{1:t}^i), a_t^i, o_{t+1}^i)$.

To prove this result, we need to demonstrate a procedure that correctly computes the new belief given only the old belief and the new action and observation. The filtering algorithm we will present in Sec. 4 achieves this goal by representing the nested belief with a finite structure that can be used to generate the infinite sequence, and showing how these structures are updated over time.

## 3.3 Extended Example: The Tiger Communication World

We now describe a simple two-agent "tiger world" where the optimal policies require the agents to coordinate their actions. In this world there are two doors: behind one randomly chosen door is a hungry tiger, and behind the other is a pile of gold. Each agent has unique abilities. Agent $l$ (the tiger listener) can hear the tiger roar, which is a noisy indication of its current location, but cannot open the doors. Agent $d$ (the door opener) can open doors but cannot hear the roars. To facilitate communication, agent $l$ has two actions, signal left and signal right, which each produce a unique observation for agent $d$. When a door is opened, the world resets and the tiger is placed behind a randomly chosen door. To act optimally, agent $l$ must listen to the tiger's roars until it is confident about the tiger's location and then send the appropriate signal to agent $d$. Agent $d$ must wait for this

| $b^{l,*}$ | $a^l$ | $\pi^l(a^l\|b^{l,*})$ | | $b^{d,*}$ | $a^d$ | $\pi^d(a^d\|b^{d,*})$ |
|---|---|---|---|---|---|---|
| $b^{l,0}(TL) > 0.8$ | $SL$ | 1.0 | | $b^{d,0}(TL) > 0.8$ | $OR$ | 1.0 |
| $b^{l,0}(TR) > 0.8$ | $SR$ | 1.0 | | $b^{d,0}(TR) > 0.8$ | $OL$ | 1.0 |
| otherwise | $L$ | 1.0 | | otherwise | $L$ | 1.0 |

Figure 1: Deterministic policies for the tiger world that depend on each agent's beliefs about the physical state, where the tiger can be on the left ($TL$) or the right ($TR$). The tiger listener, agent $l$, will signal left ($SL$) or right ($SR$) if it confident of the tiger's location. The door opener, agent $d$, will open the appropriate door when it is confident about the tiger's location. Otherwise both agents listen (to the tiger or for a signal).

signal and then open the appropriate door. Fig. 1 shows a pair of policies that achieve this desired interaction and depend only on each agent's level-zero beliefs about the state of the world. However, as we will see, the agents cannot maintain their level-zero beliefs in isolation. To correctly update these beliefs, each agent must reason about the unseen actions and observations of the other agent.

Consider the beliefs that each agent must maintain to execute its policies during a typical scenario. Assume the tiger starts behind the left door. Initially, both agents have uniform beliefs about the location of the tiger. As agent $d$ waits for a signal, it does not gain any information about the tiger's location. However, it maintains a representation of the possible beliefs for agent $l$ and knows that $l$ is receiving observations that correlate with the state of the tiger. In this case, the most likely outcome is that agent $l$ will hear enough roars on the left to do a "signal left" action. This action produces an observation for agent $d$ which allows it to gain information about $l$'s beliefs. Because agent $d$ has maintained the correspondence between the true state and agent $l$'s beliefs, it can now infer that the tiger is more likely to be on the left (it is unlikely that $l$ could have come to believe the tiger was on the left if that were not true). This inference makes agent $d$ confident enough about the tiger's location to open the right door and reset the world. Agent $l$ must also represent agent $d$'s beliefs, because it never receives any observations that indicate what actions agent $d$ is taking. It must track agent $d$'s belief updates to know that $d$ will wait for a signal and then immediately open a door. Without this information, $l$ cannot predict when the world will be reset, and thus when it should disregard past observations about the location of the tiger.

Even in this simple tiger world, we see a complicated reasoning pattern: the agents must track each others' beliefs. To update its belief about the external world, each agent must infer what actions the other agent has taken, which requires maintaining that agent's beliefs about the world. Moreover, updating the other agent's beliefs requires maintaining what it believes you believe. Continuing this reasoning to deeper levels leads to the infinitely nested beliefs defined in Sec. 3.2. However, we will never explicitly construct these infinite beliefs. Instead, we maintain a finite structure that is sufficient to recreate them to arbitrary depth, and only expand as necessary to compute action probabilities.

## 4 Efficient Filtering

In this section, we present an algorithm for performing belief updates $b_t^{i,*} = BE(b_{t-1}^{i,*}, a_{t-1}^i, o_t^i)$ on nested beliefs. This algorithm is applicable in the cooperative setting where there are commonly known policies $\pi^j(a_t^j|b_t^{j,*})$ for each agent $j$. The approach, which we call the SDS filter, maintains a set of Sparse Distributions over Sequences of past states, actions, and observations.

**Sequence distributions.** The SDS filter deals with two kinds of sequences: histories $h_{0:t}^j = (a_{0:t-1}^j, o_{1:t}^j)$ and *trajectories* $x_{0:t} = (s_{0:t}, \vec{a}_{0:t-1})$. A history represents what agent $j$ knows before acting at time $t$; a trajectory is a trace of the states and joint actions through time $t$. The filter for agent $i$ maintains the following *sequence sets*: a set $X$ of trajectories that might have occurred so far, and for each agent $j$ (including $i$ itself), a set $H^j$ of possible histories. One of the elements of $H^i$ is marked as being the history that $i$ has actually experienced. The SDS filter maintains belief information in the form of *sequence distributions* $\alpha^j(x_{0:t}|h_{0:t}^j) = p(x_{0:t}|h_{0:t}^j)$ and $\beta^j(h_{0:t}^j|x_{0:t}) = p(h_{0:t}^j|x_{0:t})$ for all agents $j$, histories $h_{0:t}^j \in H^j$, and trajectories $x_{0:t} \in X$.[2] The $\alpha^j$ distributions represent what agent $j$ would believe about the possible sequences of states and other agents' actions given $h_{0:t}^j$. The $\beta^j$ distributions represent the probability of $j$ receiving the observations in $h_{0:t}^j$ if the trajectory $x_{0:t}$ had actually happened.

The insight behind the SDS filter is that these sequence distributions can be used to compute the nested belief functions $B^{i,n}(h^i_{0:t})$ from Sec. 3.2 to arbitrary depth. The main challenge is that sets of possible histories and trajectories grow exponentially with the time $t$. To avoid this blow-up, the SDS filter does not maintain the complete set of possible sequences. We will see that some sequences can be discarded without affecting the results of the belief computations. If this pruning is insufficient, the SDS filter can drop low-probability sequences and perform approximate filtering.

A second challenge is that if we represent each sequence explicitly, the space required grows linearly with $t$. However, the belief computations do not require the details of each trajectory and history. To compute beliefs about current and future states, it suffices to maintain the sequence distributions $\alpha^j$ and $\beta^j$ defined above, along with the final state $s_t$ in each trajectory. The SDS filter maintains only this information.[3] For clarity, we will continue to use full sequence notation in the paper.

In the rest of this section, we first show how the sequence distributions can be used to compute nested beliefs of arbitrary depth. Then, we show how to maintain the sequence distributions. Finally, we present an algorithm that computes these distributions while maintaining small sequence sets.

The nested beliefs from Sec. 2.2 can be written in terms of the sequence distributions as follows:

$$B^{j,0}(h^j_{0:t})(s) = \sum_{x_{0:t} \in X \,:\, x_t = s} \alpha^j(x_{0:t}|h^j_{0:t}) \tag{2}$$

$$B^{j,n}(h^j_{0:t})(s, b^{-j,n-1}) = \sum_{x_{0:t} \in X \,:\, x_t = s} \alpha^j(x_{0:t}|h^j_{0:t}) \prod_{k \neq j} \sum_{h^k_{0:t} \in H^k} \beta^k(h^k_{0:t}|x_{0:t}) \delta(b^{k,n-1}, B^{k,n-1}(h^k_{0:t})) \tag{3}$$

At level zero, we sum over the probabilities according to agent $j$ of all trajectories with the correct final state. At level $n$, we perform the same outer sum, but for each trajectory we sum the probabilities of the histories for agents $k \neq j$ that would lead to the beliefs we are interested in. Thus, the sequence distributions at time $t$ are sufficient for computing any desired element of the infinite belief sequence $B^{j,*}(h^j_{0:t})$ for any agent $j$ and history $h^j_{0:t}$.

**Updating the distributions.** The sequence distributions are updated at each time step $t$ as follows. For each agent $j$, trajectory $x_{0:t} = (s_{0:t}, \vec{a}_{0:t-1})$ and history $h^j_{0:t} = (a^j_{0:t-1}, o^j_{1:t})$:

$$\beta^j(h^j_{0:t}|x_{0:t}) = \beta^j(h^j_{0:t-1}|x_{0:t-1})p(o^j_t|s_t, \vec{a}_{t-1}) \tag{4}$$

$$\alpha^j(x_{0:t}|h^j_{0:t}) = \alpha^j(x_{0:t-1}|h^j_{0:t-1})p(\vec{a}_{t-1}|x_{0:t-1})p(s_t|s_{t-1}, o^j_t, \vec{a}_{t-1}) \tag{5}$$

The values of $\beta^j$ on length-$t$ histories are computed from existing $\beta^j$ values by multiplying in the probability of the most recent observation. To extend $\alpha^j$ to length-$t$ trajectories, we multiply in the probability of the state transition and the probability of the agents' actions given the past trajectory:

$$p(\vec{a}_{t-1}|x_{0:t-1}) = \prod_k \sum_{h^k_{0:t-1}} \beta^k(h^k_{0:t-1}|x_{0:t-1})\pi^k(a^k_{t-1}|B^{k,*}(h^k_{0:t-1})) \tag{6}$$

Here, to predict the actions for agent $k$, we take an expectation over its possible histories $h^k_{0:t-1}$ (according to the $\beta^k$ distribution from the previous time step) of the probability of each action $a^k_{t-1}$ given the beliefs $B^{k,*}(h^k_{0:t-1})$ induced by the history. In practice, only some of the entries in $B^{k,*}(h^k_{0:t-1})$ will be needed to compute $k$'s action; for example, in the tiger world, the policies are functions of the zero-level beliefs. The necessary entries are computed from the the previous $\alpha$ and $\beta$ distributions as described in Eqs. 2 and 3. This computation is not prohibitive because, as we will see later, we only consider a small subset of the possible histories.

Returning to the example tiger world, we can see that maintaining these sequence distributions will allow us to achieve the desired interactions described in Sec. 3.3. For example, when the door opener receives a "signal left" observation, it will infer that the tiger is on the left because it has done the reasoning in Eq. 6 and determined that, with high probability, the trajectories that would have led the tiger listener to take this action are the ones where the tiger is actually on the left.

**Initialization.** Input: Distribution $p(s)$ over states.
1. Initialize trajectories and histories: $X = \{((s), ())|s \in S\}$, $H^j = \{((), ())\}$
2. Initialize distributions: $\forall x = ((s), ()) \in X, j, h^j \in H^j$: $\alpha^j(x|h^j) = p(s)$ and $\beta^j(h^j|x) = 1$.

**Filtering.** Input: Action $a_{t-1}^i$ and observation $o_t^i$.
1. Compute new sequence sets $X$ and $H^j$, for all agents $j$, by adding all possible states, actions, and observations to sequences in the previous sets. Compute new sequence distributions $\alpha^j$ and $\beta^j$, for all agents $j$, as described in Eqs. 5, 4, and 6. Mark the observed history $h_{0:t}^i \in H^i$.
2. Merge and drop sequences:
   (a) Drop trajectories and histories that are commonly known to be impossible:
      - $\forall x_{0:t} \in X$ s.t. $\forall j, h_{0:t}^j \in H^j$ . $\alpha^j(x_{0:t}|h_{0:t}^j) = 0$: Set $X = X \setminus \{x_{0:t}\}$.
      - $\forall j, h_{0:t}^j \in H^j$ s.t. $\forall x_{0:t} \in X$ . $\beta^j(h_{0:t}^j|x_{0:t}) = 0$: Set $H^j = H^j \setminus \{h_{0:t}^j\}$.
   (b) Merge histories that lead to the same beliefs:
      - $\forall j, h_{0:t}^j \in H^j, h_{0:t}'^j \in H^j$ s.t. $\forall x_{0:t} \in X$ . $\alpha^j(x_{0:t}|h_{0:t}^j) = \alpha^j(x_{0:t}|h_{0:t}'^j)$:
        Set $H^j = H^j \setminus \{h_{0:t}'^j\}$ and $\beta^j(h_{0:t}^j|x_{0:t}) = \beta^j(h_{0:t}^j|x_{0:t}) + \beta^j(h_{0:t}'^j|x_{0:t})$ for all $x_{0:t}$.
   (c) Reset when marginal of $s_t$ is common knowledge:
      - If $\forall j, k, h_{0:t}^j \in H^j, h_{0:t}^k, \in H^k, s_t$ . $\alpha^j(s_t|h_{0:t}^j) = \alpha^k(s_t|h_{0:t}^k)$:
        Reinitialize the filter using the distribution $\alpha^j(s_t|h_{0:t}^j)$ instead of the prior $p_0(s)$.
3. Prune: For all $\alpha^j$ or $\beta^j$ with $m \geq N$ non-zero entries:
   Remove the $m - N$ lowest-probability sequences and renormalize.

Figure 2: The SDS filter for agent $i$. At all times $t$, the filter maintains sequence sets $X$ and $H^j$, for all agents $j$, along with the sequence distributions $\alpha^j$ and $\beta^j$ for all agents $j$. Agent $i$'s actual observed history is marked as a distinguished element $h_{0:t}^i \in H^i$ and used to compute its beliefs $B^{i,*}(h_{0:t}^i)$.

**Filtering algorithm.** We now consider the challenge of maintaining small sequence sets. Fig. 2 provides a detailed description of the SDS filtering algorithm for agent $i$. The filter is initialized with empty histories for each agent and trajectories with single states that are distributed according to the prior. At each time $t$, Step 1 extends the sequence sets, computes the sequence distributions, and records agent $i$'s history. Running a filter with only this step would generate all possible sequences.

Step 2 introduces three operations that reduce the size of the sequence sets while guaranteeing that Eqs. 2 and 3 still produce the correct nested beliefs at time $t$. Step 2(a) removes trajectories and histories when all the agents agree that they are impossible; there is no reason to track them. For example, in the tiger communication world, the policies are such that for the first few time steps each agent will always listen (to the tiger or for signals). During this period all the trajectories where other actions are taken are known to be impossible and can be ignored. Step 2(b) merges histories for an agent $j$ that lead to the same beliefs. This is achieved by arbitrarily selecting one history to be deleted and adding its $\beta^j$ probability to the other's $\beta^j$. For example, as the tiger listener hears roars, any two observation sequences with the same numbers of roars on the left and right provide the same information about the tiger and can be merged. Step 2(c) resets the filter if the marginal over states at time $t$ has become commonly known to all the agents. For example, when both agents know that a door has been opened, this implies that the world has reset and all previous trajectories and histories can be discarded. This type of agreement is not limited to cases where the state of the world is reset. It occurs with any distribution over states that the agents agree on, for example when they localize and both know the true state, even if they disagree about the trajectory of past states.

Together, these three operators can significantly reduce the size of the sequence sets. We will see in the experiments (Sec. 5) that they enable the SDS filter to exactly track the tiger communication world extremely efficiently. However, in general, there is no guarantee that these operators will be enough to maintain small sets of trajectories and histories. Step 3 introduces an approximation by removing low-probability sequences and normalizing the belief distributions. This does guarantee that we will maintain small sequence sets, possibly at the cost of accuracy. In many domains we can ignore unlikely histories and trajectories without significantly changing the current beliefs.

## 5   Evaluation

In this section, we describe the performance of the SDS algorithm on three nested filtering problems.

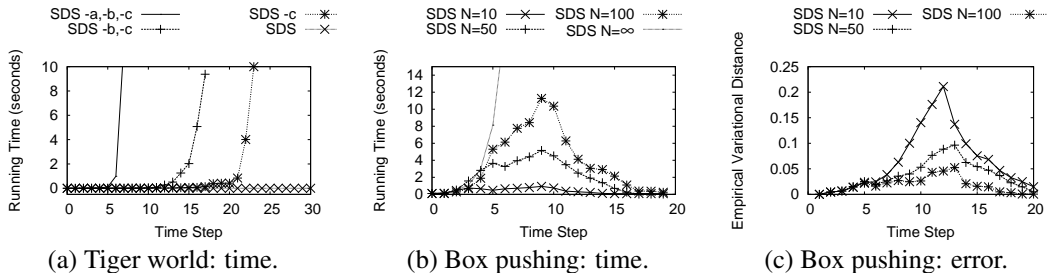

Figure 3: Time per filtering step, and error, for the SDS algorithm on two domains.

**Tiger Communication World.** The tiger communication world was described in detail in Sec. 3.3. Fig. 3(a) shows the average computation time used for filtering at each time step. The full algorithm (SDS) maintains a compact, exact representation without any pruning and takes only a fraction of a second to do each update. The graph also shows the results of disabling different parts of Step 2(a-c) of the algorithm (for example, SDS -a,-b,-c does not do any simplifications from Step 2). Without these steps, the algorithm runs in exponential time. Each simplification allows the algorithm to perform better, but all are required for constant-time performance. Since the SDS filter runs without the pruning in Step 3, we know that it computes the correct beliefs; there is no approximation error.[4]

**Box Pushing.** The DEC-POMDP literature includes several multi-agent domains; we evaluate SDS on the largest of them, known as the box-pushing domain [9]. In this scenario, two agents interact in a 3x4 grid world where they must coordinate their actions to move a large box and then independently push two small boxes. The state encodes the positions and orientations of the robots, as well as the locations of the three boxes. The agents can move forward, rotate left and right, or stay still. These actions fail with probability 0.1, leaving the state unchanged. Each agent receives deterministic observations about what is in the location in front of it (empty space, a robot, etc.). We implemented policies for each agent that consist of a set of 20 rules specifying actions given its zeroth-level beliefs about the world state. While executing their policies, the agents first coordinate to move the large box and then independently move the two small boxes. The policies are such that, with high probability, the agents will always move the boxes. There is uncertainty about when this will happen, since actions can fail. We observed, in practice, that it rarely took more than 20 steps.

Fig. 3(b) shows the running time of the SDS filter on this domain, with various pruning parameters ($N = 10, 50, 100, \infty$ in Step 3). Without pruning ($N = \infty$), the costs are too high for the filter to move beyond time step five. With pruning, however, the cost remains reasonable. Fig. 3(c) shows the error incurred with various degrees of pruning, in terms of the difference between the estimated zeroth-level beliefs for the agents and the true posterior over physical states given their observations.[5] Note that in order to accurately maintain each agent's beliefs about the physical state—which includes the position of the other robot—the filter must assign accurate probabilities to unobserved actions by the other agent , which depend on its beliefs. This is the same reasoning pattern we saw in the tiger world where we are required to maintain infinitely nested beliefs. As expected, we see that more pruning leads to faster running time but decreased accuracy. We also find that the problem is most challenging around time step ten and becomes easier in the limit, as the world moves towards the absorbing state where both agents have finished their tasks. With $N = 100$, we get high-quality estimates in an acceptable amount of time.

**Noisy Muddy Children.** The muddy children problem is a classic puzzle often discussed by researchers in epistemic logic [4]. There are $n$ agents and $2^n$ possible states. Each agent's forehead can be either muddy or clean, but it does not get any direct observations about this fact. Initially, it is commonly known that at least one agent has a muddy forehead. As time progresses, the agents follow a policy of raising their hand if they know that their forehead is muddy; they must come to this conclusion given only observations about the cleanliness of the other agents' foreheads and who has

raised their hands (this yields $2^{2n}$ possible observations for each agent). This puzzle is represented in our framework as follows. The initial knowledge is encoded with a prior that is uniform over all states with in which at least one agent is muddy. The state of the world never changes. Observations about the muddiness of the other agents are only correct with probability $\nu$, and each agent raises its hand if it assigns probability at least 0.8 to being muddy.

When there is no noise, $\nu = 1.0$, the agents behave as follows. With $m \leq n$ muddy agents, everyone waits $m$ time steps and then all of the muddy agents simultaneously raise their hands.[6] The SDS filter exhibits exactly this behavior and runs in reasonable time, using only a few seconds per filtering step, for problem instances with up to 10 agents without pruning. We also ran the filter on instances with noise ($\nu = 0.9$) and up to 5 agents. This required pruning histories to cope with the extremely large number of possible but unlikely observation sequences. The observed behavior is similar to the deterministic case: eventually, all of the $m$ muddy agents raise their hands. In expectation, this happens at a time step greater than $m$, since the agents must receive multiple observations before they are confident about each other's cleanliness. If one agent raises its hand before the others, this provides more information to the uncertain agents, who usually raise their hands soon after.

## 6 Conclusions

We have considered the problem of efficient belief update in multi-agent scenarios. We introduced the SDS algorithm, which maintains a finite belief representation that can be used to compute an infinite sequence of nested beliefs about the physical world and the beliefs of other agents. We demonstrated that on some problems, SDS can maintain this representation exactly in constant time per filtering step. On more difficult examples, SDS maintains constant-time filtering by pruning low-probability trajectories, yielding acceptable levels of approximation error.

These results show that efficient filtering is possible in multi-agent scenarios where the agents' policies are expressed as functions of their beliefs, rather than their entire observation histories. These belief-based policies are independent of the current time step, and have the potential to be more compact than history-based policies. In the single-agent setting, many successful POMDP planning algorithms construct belief-based policies; we plan to investigate how to do similar belief-based planning in the multi-agent case.

## Footnotes

[1]This is the same type of world model that is used to define POSGs and DEC-POMDPs. Since we focus on filtering instead of planning, we do not need to define reward functions for the agents.

[2] Actions are included in both histories and trajectories; when $x_{0:t}$ and $h_{0:t}^j$ specify different actions, both $\alpha^j(x_{0:t}|h_{0:t}^j)$ and $\beta^j(h_{0:t}^j|x_{0:t})$ are zero.

[3]This data structure is closely related to probabilistic Kripke structures [3] which are known to be sufficient for recreating nested beliefs. We are not aware of previous work that guarantees compactness through time.

[4]The exact version of SDS also runs in constant time on the broadcast channel domain of Hansen *et al.* [6].

[5]Because the box-pushing problem is too large for beliefs to be computed exactly, we compare the filter's performance to empirical distributions obtained by generating 10,000 sequences of trajectories and histories. We group the runs by the history $h_{0:t}^i$; for all histories that appear at least ten times, we compare the empirical distribution $\hat{b}_t$ of states occurring after that history to the filter's computed beliefs $\tilde{b}_t^{i,0}$, using the variational distance $VD(\hat{b}_t, \tilde{b}_t^{i,0}) = \sum_s |\hat{b}_t(s) - \tilde{b}_t^{i,0}(s)|$.

[6]This behavior can be verified by induction. If there is one muddy agent, it will see that the others are clean and raise its hand immediately. This implies that if no one raises their hand in the first round, there must be at least two muddy agents. At time two, they will both see only one other muddy agent and infer that they are muddy. The pattern follows for larger $m$.

## References

[1] D. S. Bernstein, E. Hansen, and S. Zilberstein. Bounded policy iteration for decentralized POMDPs. In *Proc. of the 19th International Joint Conference on Artificial Intelligence (IJCAI)*, 2005.

[2] A. Brandenburger and E. Dekel. Hierarchies of beliefs and common knowledge. *Journal of Economic Theory*, 59:189–198, 1993.

[3] R. Fagin and J. Y. Halpern. Reasoning about knowledge and probability. *Journal of the ACM*, 41(2):340–367, 1994.

[4] R. Fagin, J. Y. Halpern, Y. Moses, and M. Y. Vardi. *Reasoning About Knowledge*. The MIT Press, 1995.

[5] P. J. Gmytrasiewicz and P. Doshi. A framework for sequential planning in multi-agent settings. *Journal of Artificial Intelligence Research*, 24:49–79, 2005.

[6] E. A. Hansen, D. S. Bernstein, and S. Zilberstein. Dynamic programming for partially observable stochastic games. In *Proc. of the 19th National Conf, on Artificial Intelligence (AAAI)*, 2004.

[7] L. P. Kaelbling, M. L. Littman, and A. R. Cassandra. Planning and acting in partially observable stochastic domains. *Artificial Intelligence*, 101:99–134, 1998.

[8] B. Milch and D. Koller. Probabilistic models for agents' beliefs and decisions. In *Proc. 16th Conference on Uncertainty in Artificial Intelligence (UAI)*, 2000.

[9] S. Seuken and S. Zilberstein. Improved memory-bounded dynamic programming for decentralized POMDPs. In *Proc. of the 23rd Conference on Uncertainty in Artificial Intelligences (UAI)*, 2007.

[10] A. Shirazi and E. Amir. Probabilistic modal logic. In *Proc. of the 22nd National Conference on Artificial Intelligence (AAAI)*, 2007.

